# Learning Sparse Perceptrons

**Jeffrey C. Jackson**
Mathematics & Computer Science Dept.
Duquesne University
600 Forbes Ave
Pittsburgh, PA 15282
jackson@mathcs.duq.edu

**Mark W. Craven**
Computer Sciences Dept.
University of Wisconsin-Madison
1210 West Dayton St.
Madison, WI 53706
craven@cs.wisc.edu

## Abstract

We introduce a new algorithm designed to learn *sparse perceptrons* over input representations which include high-order features. Our algorithm, which is based on a hypothesis-boosting method, is able to PAC-learn a relatively natural class of target concepts. Moreover, the algorithm appears to work well in practice: on a set of three problem domains, the algorithm produces classifiers that utilize small numbers of features yet exhibit good generalization performance. Perhaps most importantly, our algorithm generates concept descriptions that are easy for humans to understand.

## 1 Introduction

Multi-layer perceptron (MLP) learning is a powerful method for tasks such as concept classification. However, in many applications, such as those that may involve scientific discovery, it is crucial to be able to *explain* predictions. Multi-layer perceptrons are limited in this regard, since their representations are notoriously difficult for humans to understand. We present an approach to learning understandable, yet accurate, classifiers. Specifically, our algorithm constructs *sparse perceptrons*, i.e., single-layer perceptrons that have relatively few non-zero weights. Our algorithm for learning sparse perceptrons is based on a new *hypothesis boosting* algorithm (Freund & Schapire, 1995). Although our algorithm was initially developed from a learning-theoretic point of view and retains certain theoretical guarantees (it PAC-learns the class of sparse perceptrons), it also works well in practice. Our experiments in a number of real-world domains indicate that our algorithm produces perceptrons that are relatively comprehensible, and that exhibit generalization performance comparable to that of backprop-trained MLP's (Rumelhart et al., 1986) and better than decision trees learned using C4.5 (Quinlan, 1993).

We contend that sparse perceptrons, unlike MLP's, are comprehensible because they have relatively few parameters, and each parameter describes a simple (i.e. linear) relationship. As evidence that sparse perceptrons are comprehensible, consider that such linear functions are commonly used to express domain knowledge in fields such as medicine (Spackman, 1988) and molecular biology (Stormo, 1987).

## 2  Sparse Perceptrons

A *perceptron* is a weighted threshold over the set of input features and over higher-order features consisting of functions operating on only a limited number of the input features. Informally, a *sparse* perceptron is any perceptron that has relatively few non-zero weights. For our later theoretical results we will need a more precise definition of sparseness which we develop now. Consider a Boolean function $f$ : $\{0,1\}^n \to \{-1,+1\}$. Let $C_k$ be the set of all conjunctions of at most $k$ of the inputs to $f$. $C_k$ includes the "conjunction" of 0 inputs, which we take as the identically 1 function. All of the functions in $C_k$ map to $\{-1,+1\}$, and every conjunction in $C_k$ occurs in both a positive sense (+1 represents true) and a negated sense (−1 represents true). Then the function $f$ is a $k$-perceptron if there is some integer $s$ such that $f(x) = \text{sign}(\sum_{i=1}^{s} h_i(x))$, where for all $i$, $h_i \in C_k$, and $\text{sign}(y)$ is undefined if $y = 0$ and is $y/|y|$ otherwise. Note that while we have not explicitly shown any weights in our definition of a $k$-perceptron $f$, integer weights are implicitly present in that we allow a particular $h_i \in C_k$ to appear more than once in the sum defining $f$. In fact, it is often convenient to think of a $k$-perceptron as a simple linear discriminant function with integer weights defined over a feature space with $O(n^k)$ features, one feature for each element of $C_k$.

We call a given collection of $s$ conjunctions $h_i \in C_k$ a $k$-perceptron *representation* of the corresponding function $f$, and we call $s$ the *size* of the representation. We define the *size* of a given $k$-perceptron function $f$ as the minimal size of any $k$-perceptron representation of $f$. An *$s$-sparse $k$-perceptron* is a $k$-perceptron $f$ such that the size of $f$ is at most $s$. We denote by $\mathcal{P}_k^n$ the set of Boolean functions over $\{0,1\}^n$ which can be represented as $k$-perceptrons, and we define $\mathcal{P}_k = \cup_n \mathcal{P}_k^n$. The subclass of $s$-sparse $k$-perceptrons is denoted by $\mathcal{P}_{k,s}$. We are also interested in the class $\mathcal{P}_{k,r}^R$ of $k$-perceptrons with real-valued weights, at most $r$ of which are non-zero.

## 3  The Learning Algorithm

In this section we develop our learning algorithm and prove certain performance guarantees. Our algorithm is based on a recent "hypothesis boosting" algorithm that we describe after reviewing some basic learning-theory terminology.

### 3.1  PAC Learning and Hypothesis Boosting

Following Valiant (1984), we say that a function class $\mathcal{F}$ (such as $\mathcal{P}_k$ for fixed $k$) is *(strongly) PAC-learnable* if there is an algorithm $\mathcal{A}$ and a polynomial function $p_1$ such that for any positive $\epsilon$ and $\delta$, any $f \in \mathcal{F}$ (the *target function*), and any probability distribution $D$ over the domain of $f$, with probability at least $1 - \delta$, algorithm $\mathcal{A}(EX(f,D),\epsilon,\delta)$ produces a function $h$ (the *hypothesis*) such that $\Pr[\Pr_D[f(x) \neq h(x)] > \epsilon] < \delta$. The outermost probability is over the random choices made by the $EX$ oracle and any random choices made by $\mathcal{A}$. Here $EX(f,D)$ denotes an oracle that, when queried, chooses a vector of input values $x$ with probability $D$ and returns the pair $\langle x, f(x) \rangle$ to $\mathcal{A}$. The learning algorithm $\mathcal{A}$ must run in time $p_1(n, s, \epsilon^{-1}, \delta^{-1})$, where $n$ is the length of the input vector to $f$ and $s$ is the size of

**AdaBoost**
**Input:** training set $S$ of $m$ examples of function $f$, weak learning algorithm WL that is $(\frac{1}{2} - \gamma)$-approximate, $\gamma$
**Algorithm:**

1. $T \leftarrow \frac{1}{2\gamma^2} \ln(m)$
2. for all $x \in S$, $w(x) \leftarrow 1/m$
3. **for** $i = 1$ to $T$ **do**
4.     for all $x \in S$, $D_i(x) \leftarrow w(x)/\sum_{j=1}^{m} w(x)$.
5.     invoke WL on $S$ and distribution $D_i$, producing weak hypothesis $h_i$
6.     $\epsilon_i \leftarrow \sum_{x.h_i(x) \neq f(x)} D_i(x)$
7.     $\beta_i \leftarrow \epsilon_i/(1 - \epsilon_i)$
8.     for all $x \in S$, if $h(x) = f(x)$ then $w(x) \leftarrow w(x) \cdot \beta_i$
9. **enddo**

**Output:** $h(x) \equiv sign\left(\sum_{i=1}^{T} -\ln(\beta_i) \cdot h_i(x)\right)$

Figure 1: The AdaBoost algorithm.

$f$; the algorithm is charged one unit of time for each call to $EX$. We sometimes call the function $h$ output by $\mathcal{A}$ an $\epsilon$-*approximator* (or *strong approximator*) to $f$ with respect to $D$. If $\mathcal{F}$ is PAC-learnable by an algorithm $\mathcal{A}$ that outputs only hypotheses in class $\mathcal{H}$ then we say that $\mathcal{F}$ is *PAC-learnable by $\mathcal{H}$*. If $\mathcal{F}$ is PAC-learnable for $\epsilon = 1/2 - 1/p_2(n, s)$, where $p_2$ is a polynomial function, then $\mathcal{F}$ is *weakly PAC-learnable*, and the output hypothesis $h$ in this case is called a *weak approximator*.

Our algorithm for finding sparse perceptrons is, as indicated earlier, based on the notion of hypothesis boosting. The specific boosting algorithm we use (Figure 1) is a version of the recent AdaBoost algorithm (Freund & Schapire, 1995). In the next section we apply AdaBoost to "boost" a weak learning algorithm for $\mathcal{P}_{k,s}$ into a strong learner for $\mathcal{P}_{k,s}$. AdaBoost is given a set $S$ of $m$ examples of a function $f : \{0,1\}^n \rightarrow \{-1, +1\}$ and a weak learning algorithm WL which takes $\epsilon = \frac{1}{2} - \gamma$ for a given $\gamma$ ($\gamma$ must be bounded by an inverse polynomial in $n$ and $s$). AdaBoost runs for $T = \ln(m)/(2\gamma^2)$ stages. At each stage it creates a probability distribution $D_i$ over the training set and invokes WL to find a weak hypothesis $h_i$ with respect to $D_i$ (note that an example oracle $EX(f, D_i)$ can be simulated given $D_i$ and $S$). At the end of the $T$ stages a final hypothesis $h$ is output; this is just a weighted threshold over the weak hypotheses $\{h_i \mid 1 \leq i \leq T\}$. If the weak learner succeeds in producing a $(\frac{1}{2} - \gamma)$-approximator at each stage then AdaBoost's final hypothesis is guaranteed to be consistent with the training set (Freund & Schapire, 1995).

## 3.2  PAC-Learning Sparse $k$-Perceptrons

We now show that sparse $k$-perceptrons are PAC learnable by real-weighted $k$-perceptrons having relatively few nonzero weights. Specifically, ignoring log factors, $\mathcal{P}_{k,s}$ is learnable by $\mathcal{P}_{k,O(s^2)}^{\mathrm{R}}$ for any constant $k$. We first show that, given a training set for any $f \in \mathcal{P}_{k,s}$, we can efficiently find a consistent $h \in \mathcal{P}_{k,O(s^2)}^{\mathrm{R}}$. This *consistency algorithm* is the basis of the algorithm we later apply to empirical learning problems. We then show how to turn the consistency algorithm into a PAC learning algorithm. Our proof is implicit in somewhat more general work by Freund (1993), although he did not actually present a learning algorithm for this class or analyze

the sample size needed to ensure $\epsilon$-approximation, as we do. Following Freund, we begin our development with the following lemma (Goldmann et al., 1992):

**Lemma 1 (Goldmann Hastad Razborov)** *For $f : \{0,1\}^n \to \{-1,+1\}$ and $H$, any set of functions with the same domain and range, if $f$ can be represented as $f(x) = \mathrm{sign}(\sum_{i=1}^{s} h_i(x))$, where $h_i \in H$, then for any probability distribution $D$ over $\{0,1\}^n$ there is some $h_i$ such that $\mathrm{Pr}_D[f(x) \neq h_i(x)] \leq \frac{1}{2} - \frac{1}{2s}$.*

If we specialize this lemma by taking $H = C_k$ (recall that $C_k$ is the set of conjunctions of at most $k$ input features of $f$) then this implies that for any $f \in \mathcal{P}_{k,s}$ and any probability distribution $D$ over the input features of $f$ there is some $h_i \in C_k$ that weakly approximates $f$ with respect to $D$. Therefore, given a training set $S$ and distribution $D$ that has nonzero weight only on instances in $S$, the following simple algorithm is a weak learning algorithm for $\mathcal{P}_k$: exhaustively test each of the $O(n^k)$ possible conjunctions of at most $k$ features until we find a conjunction that $(\frac{1}{2} - \frac{1}{2s})$-approximates $f$ with respect to $D$ (we can efficiently compute the approximation of a conjunction $h_i$ by summing the values of $D$ over those inputs where $h_i$ and $f$ agree). Any such conjunction can be returned as the weak hypothesis. The above lemma proves that if $f$ is a $k$-perceptron then this exhaustive search must succeed at finding such a hypothesis. Therefore, given a training set of $m$ examples of any $s$-sparse $k$-perceptron $f$, AdaBoost run with the above weak learner will, after $2s^2 \ln(m)$ stages, produce a hypothesis consistent with the training set. Because each stage adds one weak hypothesis to the output hypothesis, the final hypothesis will be a real-weighted $k$-perceptron with at most $2s^2 \ln(m)$ nonzero weights.

We can convert this consistency algorithm to a PAC learning algorithm as follows. First, given a finite set of functions $\mathcal{F}$, it is straightforward to show the following (see, e.g., Haussler, 1988):

**Lemma 2** *Let $\mathcal{F}$ be a finite set of functions over a domain $X$. For any function $f$ over $X$, any probability distribution $D$ over $X$, and any positive $\epsilon$ and $\delta$, given a set $S$ of $m$ examples drawn consecutively from $EX(f, D)$, where $m \geq \epsilon^{-1}(\ln \delta^{-1} + \ln|\mathcal{F}|)$, then $\mathrm{Pr}[\exists h \in \mathcal{F} \mid \forall x \in S \; f(x) = h(x) \;\&\; \mathrm{Pr}_D[f(x) \neq h(x)] > \epsilon] < \delta$, where the outer probability is over the random choices made by $EX(f, D)$.*

The consistency algorithm above finds a consistent hypothesis in $\mathcal{P}_{k,r}^{\mathrm{R}}$, where $r = 2s^2 \ln(m)$. Also, based on a result of Bruck (1990), it can be shown that $\ln|\mathcal{P}_{k,r}^{\mathrm{R}}| = O(r^2 + kr \log n)$. Therefore, ignoring log factors, a randomly-generated training set of size $\Omega(ks^4/\epsilon)$ is sufficient to guarantee that, with high probability, our algorithm will produce an $\epsilon$-approximator for any $s$-sparse $k$-perceptron target. In other words, the following is a PAC algorithm for $\mathcal{P}_{k,s}$: compute sufficiently large (but polynomial in the PAC parameters) $m$, draw $m$ examples from $EX(f, D)$ to create a training set, and run the consistency algorithm on this training set.

So far we have shown that sparse $k$-perceptrons are learnable by sparse perceptron hypotheses (with potentially polynomially-many more weights). In practice, of course, we expect that many real-world classification tasks cannot be performed exactly by sparse perceptrons. In fact, it can be shown that for certain (reasonable) definitions of "noisy" sparse perceptrons (loosely, functions that are approximated reasonably well by sparse perceptrons), the class of noisy sparse $k$-perceptrons is still PAC-learnable. This claim is based on results of Aslam and Decatur (1993), who present a noise-tolerant boosting algorithm. In fact, several different boosting algorithms could be used to learn $\mathcal{P}_{k,s}$ (e.g., Freund, 1993). We have chosen to use AdaBoost because it seems to offer significant practical advantages, particularly in terms of efficiency. Also, our empirical results to date indicate that our algorithm

works very well on difficult (presumably "noisy") real-world problems. However, one potential advantage of basing the algorithm on one of these earlier boosters instead of AdaBoost is that the algorithm would then produce a perceptron with integer weights while still maintaining the sparseness guarantee of the AdaBoost-based algorithm.

### 3.3  Practical Considerations

We turn now to the practical details of our algorithm, which is based on the consistency algorithm above. First, it should be noted that the theory developed above works over *discrete* input domains (Boolean or nominal-valued features). Thus, in this paper, we consider only tasks with discrete input features. Also, because the algorithm uses exhaustive search over all conjunctions of size $k$, learning time depends exponentially on the choice of $k$. In this study we to use $k = 2$ throughout, since this choice results in reasonable learning times.

Another implementation concern involves deciding when the learning algorithm should terminate. The consistency algorithm uses the size of the target function in calculating the number of boosting stages. Of course, such size information is not available in real-world applications, and in fact, the target function may not be exactly representable as a sparse perceptron. In practice, we use cross validation to determine an appropriate termination point. To facilitate comprehensibility, we also limit the number of boosting stages to at most the number of weights that would occur in an ordinary perceptron for the task. For similar reasons, we also modify the criteria used to select the weak hypothesis at each stage so that simple features are preferred over conjunctive features. In particular, given distribution $D$ at some stage $j$, for each $h_i \in C_k$ we compute a correlation $E_D[f \cdot h_i]$. We then multiply each high-order feature's correlation by $\frac{1}{k}$. The $h_i$ with the largest resulting correlation serves as the weak hypothesis for stage $j$.

## 4  Empirical Evaluation

In our experiments, we are interested in assessing both the generalization ability and the complexity of the hypotheses produced by our algorithm. We compare our algorithm to ordinary perceptrons trained using backpropagation (Rumelhart et al., 1986), multi-layer perceptrons trained using backpropagation, and decision trees induced using the C4.5 system (Quinlan, 1993). We use C4.5 in our experiments as a representative of "symbolic" learning algorithms. Symbolic algorithms are widely believed to learn hypotheses that are more comprehensible than neural networks. Additionally, to test the hypothesis that the performance of our algorithm can be explained solely by its use of second-order features, we train ordinary perceptrons using feature sets that include all pairwise conjunctions, as well as the ordinary features. To test the hypothesis that the performance of our algorithm can be explained by its use of relatively few weights, we consider ordinary perceptrons which have been pruned using a variant of the Optimal Brain Damage (OBD) algorithm (Le Cun et al., 1989). In our version of OBD, we train a perceptron until the stopping criteria are met, prune the weight with the smallest salience, and then iterate the process. We use a validation set to decide when to stop pruning weights. For each training set, we use cross-validation to select the number of hidden units (5, 10, 20, 40 or 80) for the MLP's, and the pruning confidence level for the C4.5 trees. We use a validation set to decide when to stop training for the MLP's.

We evaluate our algorithm using three real-world domains: the voting data set from the UC-Irvine database; a promoter data set which is a more complex superset of

Table 1: Test-set accuracy.

| domain | boosting | C4.5 | perceptrons | | | |
|---|---|---|---|---|---|---|
| | | | multi-layer | ordinary | 2nd-order | pruned |
| voting | 91.5% | 89.2% * | 92.2% | 90.8% | 89.2% * | 87.6% * |
| promoter | 92.7 | 84.4 * | 90.6 | 90.0 * | 88.7 * | 88.2 * |
| coding | 72.9 | 62.6 * | 71.6 * | 70.7 * | 69.8 * | 70.3 * |

Table 2: Hypothesis complexity (# weights).

| domain | boosting | perceptrons | | | |
|---|---|---|---|---|---|
| | | multi-layer | ordinary | 2nd-order | pruned |
| voting | 12 | 651 | 30 | 450 | 12 |
| promoters | 41 | 2267 | 228 | 25764 | 59 |
| protein coding | 52 | 4270 | 60 | 1740 | 37 |

UC-Irvine one; and a data set in which the task is to recognize protein-coding regions in DNA (Craven & Shavlik, 1993). We remove the `physician-fee-freeze` feature from the voting data set to make the problem more difficult. We conduct our experiments using a 10-fold cross validation methodology, except for in the protein-coding domain. Because of certain domain-specific characteristics of this data set, we use 4-fold cross-validation for our experiments with it.

Table 1 reports test-set accuracy for each method on all three domains. We measure the statistical significance of accuracy differences using a paired, two-tailed $t$-test. The symbol '*' marks results in cases where another algorithm is less accurate than our boosting algorithm at the $p \leq 0.05$ level of significance. No other algorithm is significantly better than our boosting method in any of the domains. From these results we conclude that (1) our algorithm exhibits good generalization performance on number of interesting real-world problems, and (2) the generalization performance of our algorithm is not explained solely by its use of second-order features, nor is it solely explained by the sparseness of the perceptrons it produces. An interesting open question is whether perceptrons trained with both pruning and second-order features are able to match the accuracy of our algorithm; we plan to investigate this question in future work.

Table 2 reports the average number of weights for all of the perceptrons. For all three problems, our algorithm produces perceptrons with fewer weights than the MLP's, the ordinary perceptrons, and the perceptrons with second-order features. The sizes of the OBD-pruned perceptrons and those produced by our algorithm are comparable for all three domains. Recall, however, that for all three tasks, the perceptrons learned by our algorithm had significantly better generalization performance than their similar-sized OBD-pruned counterparts. We contend that the sizes of the perceptrons produced by our algorithm are within the bounds of what humans can readily understand. In the biological literature, for example, linear discriminant functions are frequently used to communicate domain knowledge about sequences of interest. These functions frequently involve more weights than the perceptrons produced by our algorithm. We conclude, therefore, that our algorithm produces hypotheses that are not only accurate, but also comprehensible.

We believe that the results on the protein-coding domain are especially interesting. The input representation for this problem consists of 15 nominal features representing 15 consecutive *bases* in a DNA sequence. In the regions of DNA that encode proteins (the positive examples in our task), non-overlapping triplets of consecu-

tive bases represent meaningful "words" called *codons*. In previous work (Craven & Shavlik, 1993), it has been found that a feature set that explicitly represents codons results in better generalization than a representation of just bases. However, we used the bases representation in our experiments in order to investigate the ability of our algorithm to select the "right" second-order features. Interestingly, nearly all of the second-order features included in our sparse perceptrons represent conjunctions of bases that are in the same codon. This result suggests that our algorithm is especially good at selecting relevant features from large feature sets.

## 5   Future Work

Our present algorithm has a number of limitations which we plan to address. Two areas of current research are generalizing the algorithm for application to problems with real-valued features and developing methods for automatically suggesting high-order features to be included in our algorithm's feature set.

### Acknowledgements

Mark Craven was partially supported by ONR grant N00014-93-1-0998. Jeff Jackson was partially supported by NSF grant CCR-9119319.

## References

Aslam, J. A. & Decatur, S. E. (1993). General bounds on statistical query learning and PAC learning with noise via hypothesis boosting. In *Proc. of the 34th Annual Annual Symposium on Foundations of Computer Science*, (pp. 282–291).

Bruck, J. (1990). Harmonic analysis of polynomial threshold functions. *SIAM Journal of Discrete Mathematics*, 3(2):168–177.

Craven, M. W. & Shavlik, J. W. (1993). Learning to represent codons: A challenge problem for constructive induction. In *Proc. of the 13th International Joint Conf. on Artificial Intelligence*, (pp. 1319–1324), Chambery, France.

Freund, Y. (1993). *Data Filtering and Distribution Modeling Algorithms for Machine Learning*. PhD thesis, University of California at Santa Cruz.

Freund, Y. & Schapire, R. E. (1995). A decision-theoretic generalization of on-line learning and an application to boosting. In *Proc. of the 2nd Annual European Conf. on Computational Learning Theory*.

Goldmann, M., Hastad, J., & Razborov, A. (1992). Majority gates vs. general weighted threshold gates. In *Proc. of the 7th IEEE Conf. on Structure in Complexity Theory*.

Haussler, D. (1988). Quantifying inductive bias: AI learning algorithms and Valiant's learning framework. *Artificial Intelligence*, (pp. 177–221).

Le Cun, Y., Denker, J. S., & Solla, S. A. (1989). Optimal brain damage. In Touretzky, D., editor, *Advances in Neural Information Processing Systems (volume 2)*.

Quinlan, J. R. (1993). *C4.5: Programs for Machine Learning*. Morgan Kaufmann.

Rumelhart, D., Hinton, G., & Williams, R. (1986). Learning internal representations by error propagation. In Rumelhart, D. & McClelland, J., editors, *Parallel Distributed Processing: Explorations in the microstructure of cognition. Volume 1*. MIT Press.

Spackman, K. A. (1988). Learning categorical decision criteria. In *Proc. of the 5th International Conf. on Machine Learning*, (pp. 36–46), Ann Arbor, MI.

Stormo, G. (1987). Identifying coding sequences. In Bishop, M. J. & Rawlings, C. J., editors, *Nucleic Acid and Protein Sequence Analysis: A Practical Approach*. IRL Press.

Valiant, L. G. (1984). A theory of the learnable. *Comm. of the ACM*, 27(11):1134–1142.